# Kernel Methods for Deep Learning

**Youngmin Cho and Lawrence K. Saul**
Department of Computer Science and Engineering
University of California, San Diego
9500 Gilman Drive, Mail Code 0404
La Jolla, CA 92093-0404
{yoc002,saul}@cs.ucsd.edu

## Abstract

We introduce a new family of positive-definite kernel functions that mimic the computation in large, multilayer neural nets. These kernel functions can be used in shallow architectures, such as support vector machines (SVMs), or in deep kernel-based architectures that we call multilayer kernel machines (MKMs). We evaluate SVMs and MKMs with these kernel functions on problems designed to illustrate the advantages of deep architectures. On several problems, we obtain better results than previous, leading benchmarks from both SVMs with Gaussian kernels as well as deep belief nets.

## 1 Introduction

Recent work in machine learning has highlighted the circumstances that appear to favor deep architectures, such as multilayer neural nets, over shallow architectures, such as support vector machines (SVMs) [1]. Deep architectures learn complex mappings by transforming their inputs through multiple layers of nonlinear processing [2]. Researchers have advanced several motivations for deep architectures: the wide range of functions that can be parameterized by composing weakly nonlinear transformations, the appeal of hierarchical distributed representations, and the potential for combining unsupervised and supervised methods. Experiments have also shown the benefits of deep learning in several interesting applications [3, 4, 5].

Many issues surround the ongoing debate over deep versus shallow architectures [1, 6]. Deep architectures are generally more difficult to train than shallow ones. They involve difficult nonlinear optimizations and many heuristics. The challenges of deep learning explain the early and continued appeal of SVMs, which learn nonlinear classifiers via the "kernel trick". Unlike deep architectures, SVMs are trained by solving a simple problem in quadratic programming. However, SVMs cannot seemingly benefit from the advantages of deep learning.

Like many, we are intrigued by the successes of deep architectures yet drawn to the elegance of kernel methods. In this paper, we explore the possibility of deep learning in kernel machines. Though we share a similar motivation as previous authors [7], our approach is very different. Our paper makes two main contributions. First, we develop a new family of kernel functions that mimic the computation in large neural nets. Second, using these kernel functions, we show how to train multilayer kernel machines (MKMs) that benefit from many advantages of deep learning.

The organization of this paper is as follows. In section 2, we describe a new family of kernel functions and experiment with their use in SVMs. Our results on SVMs are interesting in their own right; they also foreshadow certain trends that we observe (and certain choices that we make) for the MKMs introduced in section 3. In this section, we describe a kernel-based architecture with multiple layers of nonlinear transformation. The different layers are trained using a simple combination of supervised and unsupervised methods. Finally, we conclude in section 4 by evaluating the strengths and weaknesses of our approach.

## 2 Arc-cosine kernels

In this section, we develop a new family of kernel functions for computing the similarity of vector inputs $\mathbf{x}, \mathbf{y} \in \Re^d$. As shorthand, let $\Theta(z) = \frac{1}{2}(1 + \mathrm{sign}(z))$ denote the Heaviside step function. We define the $n$th order *arc-cosine* kernel function via the integral representation:

$$k_n(\mathbf{x}, \mathbf{y}) \;=\; 2 \int d\mathbf{w} \; \frac{e^{-\frac{\|\mathbf{w}\|^2}{2}}}{(2\pi)^{d/2}} \; \Theta(\mathbf{w} \cdot \mathbf{x}) \, \Theta(\mathbf{w} \cdot \mathbf{y}) \, (\mathbf{w} \cdot \mathbf{x})^n \, (\mathbf{w} \cdot \mathbf{y})^n \tag{1}$$

The integral representation makes it straightforward to show that these kernel functions are positive-semidefinite. The kernel function in eq. (1) has interesting connections to neural computation [8] that we explore further in sections 2.2–2.3. However, we begin by elucidating its basic properties.

### 2.1 Basic properties

We show how to evaluate the integral in eq. (1) analytically in the appendix. The final result is most easily expressed in terms of the angle $\theta$ between the inputs:

$$\theta \;=\; \cos^{-1}\left(\frac{\mathbf{x} \cdot \mathbf{y}}{\|\mathbf{x}\|\|\mathbf{y}\|}\right). \tag{2}$$

The integral in eq. (1) has a simple, trivial dependence on the magnitudes of the inputs $\mathbf{x}$ and $\mathbf{y}$, but a complex, interesting dependence on the angle between them. In particular, we can write:

$$k_n(\mathbf{x}, \mathbf{y}) \;=\; \frac{1}{\pi} \, \|\mathbf{x}\|^n \|\mathbf{y}\|^n J_n(\theta) \tag{3}$$

where all the angular dependence is captured by the family of functions $J_n(\theta)$. Evaluating the integral in the appendix, we show that this angular dependence is given by:

$$J_n(\theta) \;=\; (-1)^n (\sin\theta)^{2n+1} \left(\frac{1}{\sin\theta} \frac{\partial}{\partial\theta}\right)^n \left(\frac{\pi - \theta}{\sin\theta}\right). \tag{4}$$

For $n = 0$, this expression reduces to the supplement of the angle between the inputs. However, for $n > 0$, the angular dependence is more complicated. The first few expressions are:

$$J_0(\theta) \;=\; \pi - \theta \tag{5}$$

$$J_1(\theta) \;=\; \sin\theta + (\pi - \theta)\cos\theta \tag{6}$$

$$J_2(\theta) \;=\; 3\sin\theta\cos\theta + (\pi - \theta)(1 + 2\cos^2\theta) \tag{7}$$

We describe eq. (3) as an arc-cosine kernel because for $n = 0$, it takes the simple form $k_0(\mathbf{x}, \mathbf{y}) = 1 - \frac{1}{\pi}\cos^{-1}\frac{\mathbf{x} \cdot \mathbf{y}}{\|\mathbf{x}\|\|\mathbf{y}\|}$. In fact, the zeroth and first order kernels in this family are strongly motivated by previous work in neural computation. We explore these connections in the next section.

Arc-cosine kernels have other intriguing properties. From the magnitude dependence in eq. (3), we observe the following: (i) the $n = 0$ arc-cosine kernel maps inputs $\mathbf{x}$ to the unit hypersphere in feature space, with $k_0(\mathbf{x}, \mathbf{x}) = 1$; (ii) the $n = 1$ arc-cosine kernel preserves the norm of inputs, with $k_1(\mathbf{x}, \mathbf{x}) = \|\mathbf{x}\|^2$; (iii) higher order ($n > 1$) arc-cosine kernels expand the dynamic range of the inputs, with $k_n(\mathbf{x}, \mathbf{x}) \sim \|\mathbf{x}\|^{2n}$. Properties (i)–(iii) are shared respectively by radial basis function (RBF), linear, and polynomial kernels. Interestingly, though, the $n = 1$ arc-cosine kernel is highly nonlinear, also satisfying $k_1(\mathbf{x}, -\mathbf{x}) = 0$ for all inputs $\mathbf{x}$. As a practical matter, we note that arc-cosine kernels do not have any continuous tuning parameters (such as the kernel width in RBF kernels), which can be laborious to set by cross-validation.

### 2.2 Computation in single-layer threshold networks

Consider the single-layer network shown in Fig. 1 (left) whose weights $W_{ij}$ connect the $j$th input unit to the $i$th output unit. The network maps inputs $\mathbf{x}$ to outputs $\mathbf{f}(\mathbf{x})$ by applying an elementwise nonlinearity to the matrix-vector product of the inputs and the weight matrix: $\mathbf{f}(\mathbf{x}) = g(\mathbf{W}\mathbf{x})$. The nonlinearity is described by the network's so-called activation function. Here we consider the family of one-sided polynomial activation functions $g_n(z) = \Theta(z)z^n$ illustrated in the right panel of Fig. 1.

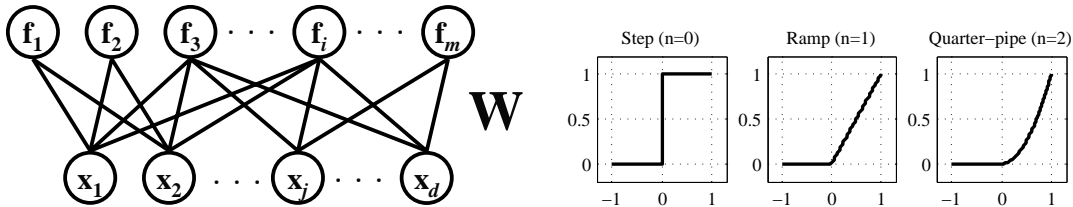

Figure 1: Single layer network and activation functions

For $n=0$, the activation function is a step function, and the network is an array of perceptrons. For $n=1$, the activation function is a ramp function (or rectification nonlinearity [9]), and the mapping $\mathbf{f}(\mathbf{x})$ is piecewise linear. More generally, the nonlinear (non-polynomial) behavior of these networks is induced by thresholding on weighted sums. We refer to networks with these activation functions as single-layer threshold networks of degree $n$.

Computation in these networks is closely connected to computation with the arc-cosine kernel function in eq. (1). To see the connection, consider how inner products are transformed by the mapping in single-layer threshold networks. As notation, let the vector $\mathbf{w}_i$ denote $i$th row of the weight matrix $\mathbf{W}$. Then we can express the inner product between different outputs of the network as:

$$\mathbf{f}(\mathbf{x}) \cdot \mathbf{f}(\mathbf{y}) \;=\; \sum_{i=1}^{m} \Theta(\mathbf{w}_i \cdot \mathbf{x}) \Theta(\mathbf{w}_i \cdot \mathbf{y}) (\mathbf{w}_i \cdot \mathbf{x})^n (\mathbf{w}_i \cdot \mathbf{y})^n, \tag{8}$$

where $m$ is the number of output units. The connection with the arc-cosine kernel function emerges in the limit of very large networks [10, 8]. Imagine that the network has an infinite number of output units, and that the weights $W_{ij}$ are Gaussian distributed with zero mean and unit variance. In this limit, we see that eq. (8) reduces to eq. (1) up to a trivial multiplicative factor: $\lim_{m \to \infty} \frac{2}{m} \mathbf{f}(\mathbf{x}) \cdot \mathbf{f}(\mathbf{y}) = k_n(\mathbf{x}, \mathbf{y})$. Thus the arc-cosine kernel function in eq. (1) can be viewed as the inner product between feature vectors derived from the mapping of an infinite single-layer threshold network [8].

Many researchers have noted the general connection between kernel machines and neural networks with one layer of hidden units [1]. The $n=0$ arc-cosine kernel in eq. (1) can also be derived from an earlier result obtained in the context of Gaussian processes [8]. However, we are unaware of any previous theoretical or empirical work on the general family of these kernels for degrees $n \geq 0$.

Arc-cosine kernels differ from polynomial and RBF kernels in one especially interesting respect. As highlighted by the integral representation in eq. (1), arc-cosine kernels induce feature spaces that mimic the sparse, nonnegative, distributed representations of single-layer threshold networks. Polynomial and RBF kernels do not encode their inputs in this way. In particular, the feature vector induced by polynomial kernels is neither sparse nor nonnegative, while the feature vector induced by RBF kernels resembles the localized output of a soft vector quantizer. Further implications of this difference are explored in the next section.

## 2.3 Computation in multilayer threshold networks

A kernel function can be viewed as inducing a nonlinear mapping from inputs $\mathbf{x}$ to feature vectors $\mathbf{\Phi}(\mathbf{x})$. The kernel computes the inner product in the induced feature space: $k(\mathbf{x}, \mathbf{y}) = \mathbf{\Phi}(\mathbf{x}) \cdot \mathbf{\Phi}(\mathbf{y})$. In this section, we consider how to compose the nonlinear mappings induced by kernel functions. Specifically, we show how to derive new kernel functions

$$k^{(\ell)}(\mathbf{x}, \mathbf{y}) \;=\; \underbrace{\mathbf{\Phi}(\mathbf{\Phi}(...\mathbf{\Phi}(\mathbf{x})))}_{\ell \text{ times}} \cdot \underbrace{\mathbf{\Phi}(\mathbf{\Phi}(...\mathbf{\Phi}(\mathbf{y})))}_{\ell \text{ times}} \tag{9}$$

which compute the inner product after $\ell$ successive applications of the nonlinear mapping $\mathbf{\Phi}(\cdot)$. Our motivation is the following: intuitively, if the base kernel function $k(\mathbf{x}, \mathbf{y}) = \mathbf{\Phi}(\mathbf{x}) \cdot \mathbf{\Phi}(\mathbf{y})$ mimics the computation in a single-layer network, then the iterated mapping in eq. (9) should mimic the computation in a multilayer network.

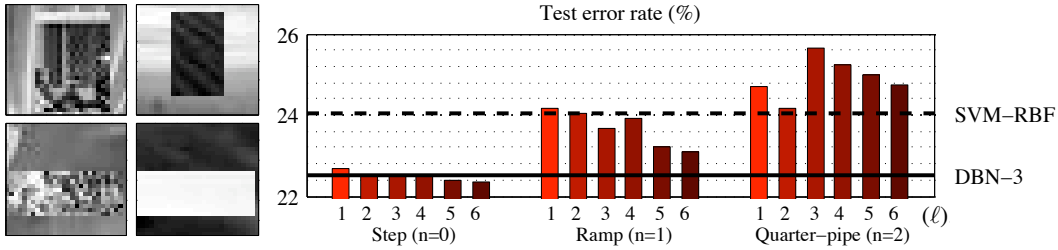

Figure 2: *Left*: examples from the *rectangles-image* data set. *Right*: classification error rates on the test set. SVMs with arc-cosine kernels have error rates from 22.36–25.64%. Results are shown for kernels of varying degree ($n$) and levels of recursion ($\ell$). The best previous results are 24.04% for SVMs with RBF kernels and 22.50% for deep belief nets [11]. See text for details.

We first examine the results of this procedure for widely used kernels. Here we find that the iterated mapping in eq. (9) does not yield particularly interesting results. Consider the two-fold composition that maps $\mathbf{x}$ to $\mathbf{\Phi}(\mathbf{\Phi}(\mathbf{x}))$. For linear kernels $k(\mathbf{x}, \mathbf{y}) = \mathbf{x} \cdot \mathbf{y}$, the composition is trivial: we obtain the identity map $\mathbf{\Phi}(\mathbf{\Phi}(\mathbf{x})) = \mathbf{\Phi}(\mathbf{x}) = \mathbf{x}$. For homogeneous polynomial kernels $k(\mathbf{x}, \mathbf{y}) = (\mathbf{x} \cdot \mathbf{y})^d$, the composition yields:

$$\mathbf{\Phi}(\mathbf{\Phi}(\mathbf{x})) \cdot \mathbf{\Phi}(\mathbf{\Phi}(\mathbf{y})) = (\mathbf{\Phi}(\mathbf{x}) \cdot \mathbf{\Phi}(\mathbf{y}))^d = ((\mathbf{x} \cdot \mathbf{y})^d)^d = (\mathbf{x} \cdot \mathbf{y})^{d^2}. \tag{10}$$

The above result is not especially interesting: the kernel implied by this composition is also polynomial, just of higher degree ($d^2$ versus $d$) than the one from which it was constructed. Likewise, for RBF kernels $k(\mathbf{x}, \mathbf{y}) = e^{-\lambda \|\mathbf{x} - \mathbf{y}\|^2}$, the composition yields:

$$\mathbf{\Phi}(\mathbf{\Phi}(\mathbf{x})) \cdot \mathbf{\Phi}(\mathbf{\Phi}(\mathbf{y})) = e^{-\lambda \|\mathbf{\Phi}(\mathbf{x}) - \mathbf{\Phi}(\mathbf{y})\|^2} = e^{-2\lambda(1 - k(\mathbf{x}, \mathbf{y}))}. \tag{11}$$

Though non-trivial, eq. (11) does not represent a particularly interesting computation. Recall that RBF kernels mimic the computation of soft vector quantizers, with $k(\mathbf{x}, \mathbf{y}) \ll 1$ when $\|\mathbf{x} - \mathbf{y}\|$ is large compared to the kernel width. It is hard to see how the iterated mapping $\mathbf{\Phi}(\mathbf{\Phi}(\mathbf{x}))$ would generate a qualitatively different representation than the original mapping $\mathbf{\Phi}(\mathbf{x})$.

Next we consider the $\ell$-fold composition in eq. (9) for arc-cosine kernel functions. We state the result in the form of a recursion. The base case is given by eq. (3) for kernels of depth $\ell = 1$ and degree $n$. The inductive step is given by:

$$k_n^{(l+1)}(\mathbf{x}, \mathbf{y}) = \frac{1}{\pi} \left[ k_n^{(l)}(\mathbf{x}, \mathbf{x}) \, k_n^{(l)}(\mathbf{y}, \mathbf{y}) \right]^{n/2} J_n\left(\theta_n^{(\ell)}\right), \tag{12}$$

where $\theta_n^{(\ell)}$ is the angle between the images of $\mathbf{x}$ and $\mathbf{y}$ in the feature space induced by the $\ell$-fold composition. In particular, we can write:

$$\theta_n^{(\ell)} = \cos^{-1}\left( k_n^{(\ell)}(\mathbf{x}, \mathbf{y}) \left[ k_n^{(\ell)}(\mathbf{x}, \mathbf{x}) \, k_n^{(\ell)}(\mathbf{y}, \mathbf{y}) \right]^{-1/2} \right). \tag{13}$$

The recursion in eq. (12) is simple to compute in practice. The resulting kernels mimic the computations in large multilayer threshold networks. Above, for simplicity, we have assumed that the arc-cosine kernels have the same degree $n$ at every level (or *layer*) $\ell$ of the recursion. We can also use kernels of different degrees at different layers. In the next section, we experiment with SVMs whose kernel functions are constructed in this way.

## 2.4 Experiments on binary classification

We evaluated SVMs with arc-cosine kernels on two challenging data sets of $28 \times 28$ grayscale pixel images. These data sets were specifically constructed to compare deep architectures and kernel machines [11]. In the first data set, known as *rectangles-image*, each image contains an occluding rectangle, and the task is to determine whether the width of the rectangle exceeds its height; examples are shown in Fig. 2 (left). In the second data set, known as *convex*, each image contains a white region, and the task is to determine whether the white region is convex; examples are shown

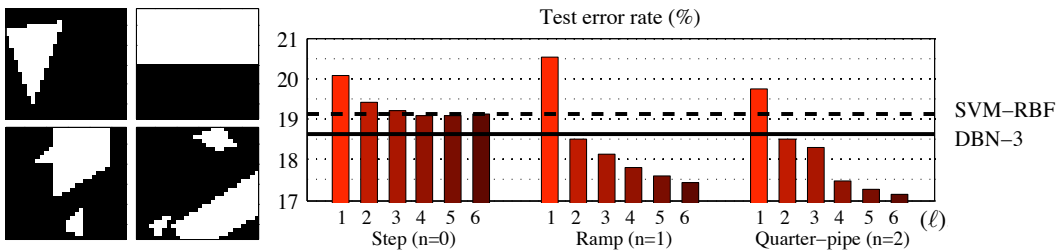

Figure 3: *Left*: examples from the *convex* data set. *Right*: classification error rates on the test set. SVMs with arc-cosine kernels have error rates from 17.15–20.51%. Results are shown for kernels of varying degree ($n$) and levels of recursion ($\ell$). The best previous results are 19.13% for SVMs with RBF kernels and 18.63% for deep belief nets [11]. See text for details.

in Fig. 3 (left). The *rectangles-image* data set has 12000 training examples, while the *convex* data set has 8000 training examples; both data sets have 50000 test examples. These data sets have been extensively benchmarked by previous authors [11]. Our experiments in binary classification focused on these data sets because in previously reported benchmarks, they exhibited the biggest performance gap between deep architectures (e.g., deep belief nets) and traditional SVMs.

We followed the same experimental methodology as previous authors [11]. SVMs were trained using libSVM (version 2.88) [12], a publicly available software package. For each SVM, we used the last 2000 training examples as a validation set to choose the margin penalty parameter; after choosing this parameter by cross-validation, we then retrained each SVM using all the training examples. For reference, we also report the best results obtained previously from three-layer deep belief nets (DBN-3) and SVMs with RBF kernels (SVM-RBF). These references appear to be representative of the current state-of-the-art for deep and shallow architectures on these data sets.

Figures 2 and 3 show the test set error rates from arc-cosine kernels of varying degree ($n$) and levels of recursion ($\ell$). We experimented with kernels of degree $n = 0, 1$ and 2, corresponding to threshold networks with "step", "ramp", and "quarter-pipe" activation functions. We also experimented with the multilayer kernels described in section 2.3, composed from one to six levels of recursion. Overall, the figures show that many SVMs with arc-cosine kernels outperform traditional SVMs, and a certain number also outperform deep belief nets. In addition to their solid performance, we note that SVMs with arc-cosine kernels are very straightforward to train; unlike SVMs with RBF kernels, they do not require tuning a kernel width parameter, and unlike deep belief nets, they do not require solving a difficult nonlinear optimization or searching over possible architectures.

Our experiments with multilayer kernels revealed that these SVMs only performed well when arc-cosine kernels of degree $n = 1$ were used at higher ($\ell > 1$) levels in the recursion. Figs. 2 and 3 therefore show only these sets of results; in particular, each group of bars shows the test error rates when a particular kernel (of degree $n = 0, 1, 2$) was used at the first layer of nonlinearity, while the $n = 1$ kernel was used at successive layers. We hypothesize that only $n = 1$ arc-cosine kernels preserve sufficient information about the magnitude of their inputs to work effectively in composition with other kernels. Recall that only the $n = 1$ arc-cosine kernel preserves the norm of its inputs: the $n = 0$ kernel maps all inputs onto a unit hypersphere in feature space, while higher-order ($n > 1$) kernels induce feature spaces with different dynamic ranges.

Finally, the results on both data sets reveal an interesting trend: the multilayer arc-cosine kernels often perform better than their single-layer counterparts. Though SVMs are (inherently) shallow architectures, this trend suggests that for these problems in binary classification, arc-cosine kernels may be yielding some of the advantages typically associated with deep architectures.

## 3 Deep learning

In this section, we explore how to use kernel methods in deep architectures [7]. We show how to train deep kernel-based architectures by a simple combination of supervised and unsupervised methods. Using the arc-cosine kernels in the previous section, these multilayer kernel machines (MKMs) perform very competitively on multiclass data sets designed to foil shallow architectures [11].

### 3.1 Multilayer kernel machines

We explored how to train MKMs in stages that involve kernel PCA [13] and feature selection [14] at intermediate hidden layers and large-margin nearest neighbor classification [15] at the final output layer. Specifically, for $\ell$-layer MKMs, we considered the following training procedure:

---
**1. Prune uninformative features from the input space.**
**2. Repeat $\ell$ times:**
    **(a) Compute principal components in the feature space induced by a nonlinear kernel.**
    **(b) Prune uninformative components from the feature space.**
**3. Learn a Mahalanobis distance metric for nearest neighbor classification.**
---

The individual steps in this procedure are well-established methods; only their combination is new. While many other approaches are worth investigating, our positive results from the above procedure provide a first proof-of-concept. We discuss each of these steps in greater detail below.

*Kernel PCA.* Deep learning in MKMs is achieved by iterative applications of kernel PCA [13]. This use of kernel PCA was suggested over a decade ago [16] and more recently inspired by the pre-training of deep belief nets by unsupervised methods. In MKMs, the outputs (or features) from kernel PCA at one layer are the inputs to kernel PCA at the next layer. However, we do not strictly transmit each layer's top principal components to the next layer; some components are discarded if they are deemed uninformative. While any nonlinear kernel can be used for the layerwise PCA in MKMs, arc-cosine kernels are natural choices to mimic the computations in large neural nets.

*Feature selection.* The layers in MKMs are trained by interleaving a supervised method for feature selection with the unsupervised method of kernel PCA. The feature selection is used to prune away uninformative features at each layer in the MKM (including the zeroth layer which stores the raw inputs). Intuitively, this feature selection helps to focus the unsupervised learning in MKMs on statistics of the inputs that actually contain information about the class labels. We prune features at each layer by a simple two-step procedure that first ranks them by estimates of their mutual information, then truncates them using cross-validation. More specifically, in the first step, we discretize each real-valued feature and construct class-conditional and marginal histograms of its discretized values; then, using these histograms, we estimate each feature's mutual information with the class label and sort the features in order of these estimates [14]. In the second step, considering only the first $w$ features in this ordering, we compute the error rates of a basic $k$NN classifier using Euclidean distances in feature space. We compute these error rates on a held-out set of validation examples for many values of $k$ and $w$ and record the optimal values for each layer. The optimal $w$ determines the number of informative features passed onto the next layer; this is essentially the *width* of the layer. In practice, we varied $k$ from 1 to 15 and $w$ from 10 to 300; though exhaustive, this cross-validation can be done quickly and efficiently by careful bookkeeping. Note that this procedure determines the architecture of the network in a greedy, layer-by-layer fashion.

*Distance metric learning.* Test examples in MKMs are classified by a variant of $k$NN classification on the outputs of the final layer. Specifically, we use large margin nearest neighbor (LMNN) classification [15] to learn a Mahalanobis distance metric for these outputs, though other methods are equally viable [17]. The use of LMNN is inspired by the supervised fine-tuning of weights in the training of deep architectures [18]. In MKMs, however, this supervised training only occurs at the final layer (which underscores the importance of feature selection in earlier layers). LMNN learns a distance metric by solving a problem in semidefinite programming; one advantage of LMNN is that the required optimization is convex. Test examples are classified by the energy-based decision rule for LMNN [15], which was itself inspired by earlier work on multilayer neural nets [19].

### 3.2 Experiments on multiway classification

We evaluated MKMs on the two multiclass data sets from previous benchmarks [11] that exhibited the largest performance gap between deep and shallow architectures. The data sets were created from the MNIST data set [20] of $28 \times 28$ grayscale handwritten digits. The *mnist-back-rand* data set was generated by filling the image background by random pixel values, while the *mnist-back-image* data set was generated by filling the image background with random image patches; examples are shown in Figs. 4 and 5. Each data set contains 12000 and 50000 training and test examples, respectively.

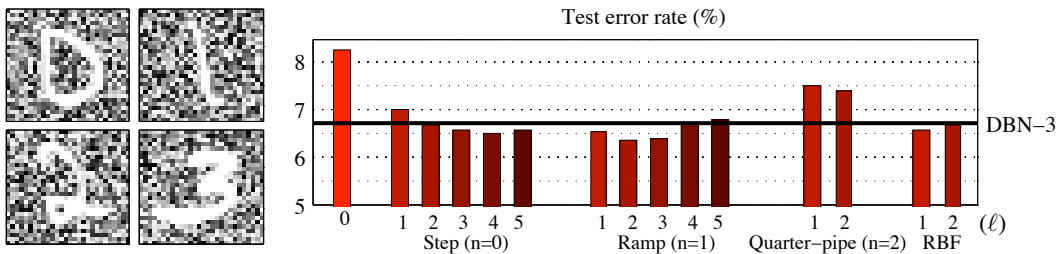

Figure 4: *Left*: examples from the *mnist-back-rand* data set. *Right*: classification error rates on the test set for MKMs with different kernels and numbers of layers $\ell$. MKMs with arc-cosine kernel have error rates from 6.36–7.52%. The best previous results are 14.58% for SVMs with RBF kernels and 6.73% for deep belief nets [11].

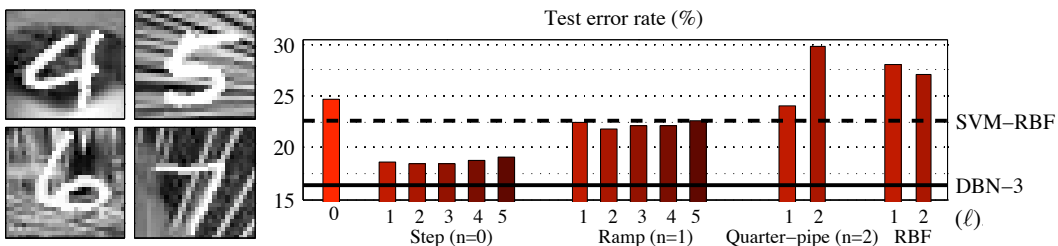

Figure 5: *Left*: examples from the *mnist-back-image* data set. *Right*: classification error rates on the test set for MKMs with different kernels and numbers of layers $\ell$. MKMs with arc-cosine kernel have error rates from 18.43–29.79%. The best previous results are 22.61% for SVMs with RBF kernels and 16.31% for deep belief nets [11].

We trained MKMs with arc-cosine kernels and RBF kernels in each layer. For each data set, we initially withheld the last 2000 training examples as a validation set. Performance on this validation set was used to determine each MKM's architecture, as described in the previous section, and also to set the kernel width in RBF kernels, following the same methodology as earlier studies [11]. Once these parameters were set by cross-validation, we re-inserted the validation examples into the training set and used all 12000 training examples for feature selection and distance metric learning. For kernel PCA, we were limited by memory requirements to processing only 6000 out of 12000 training examples. We chose these 6000 examples randomly, but repeated each experiment five times to obtain a measure of average performance. The results we report for each MKM are the average performance over these five runs.

The right panels of Figs. 4 and 5 show the test set error rates of MKMs with different kernels and numbers of layers $\ell$. For reference, we also show the best previously reported results [11] using traditional SVMs (with RBF kernels) and deep belief nets (with three layers). MKMs perform significantly better than shallow architectures such as SVMs with RBF kernels or LMNN with feature selection (reported as the case $\ell = 0$). Compared to deep belief nets, the leading MKMs obtain slightly lower error rates on one data set and slightly higher error rates on another.

We can describe the architecture of an MKM by the number of selected features at each layer (including the input layer). The number of features essentially corresponds to the number of units in each layer of a neural net. For the *mnist-back-rand* data set, the best MKM used an $n=1$ arc-cosine kernel and 300-90-105-136-126-240 features at each layer. For the *mnist-back-image* data set, the best MKM used an $n=0$ arc-cosine kernel and 300-50-130-240-160-150 features at each layer.

MKMs worked best with arc-cosine kernels of degree $n=0$ and $n=1$. The kernel of degree $n=2$ performed less well in MKMs, perhaps because multiple iterations of kernel PCA distorted the dynamic range of the inputs (which in turn seemed to complicate the training for LMNN). MKMs with RBF kernels were difficult to train due to the sensitive dependence on kernel width parameters. It was extremely time-consuming to cross-validate the kernel width at each layer of the MKM. We only obtained meaningful results for one and two-layer MKMs with RBF kernels.

We briefly summarize many results that we lack space to report in full. We also experimented on multiclass data sets using SVMs with single and multi-layer arc-cosine kernels, as described in section 2. For multiclass problems, these SVMs compared poorly to deep architectures (both DBNs and MKMs), presumably because they had no unsupervised training that shared information across examples from *all* different classes. In further experiments on MKMs, we attempted to evaluate the individual contributions to performance from feature selection and LMNN classification. Feature selection helped significantly on the *mnist-back-image* data set, but only slightly on the *mnist-back-random* data set. Finally, LMNN classification in the output layer yielded consistent improvements over basic $k$NN classification provided that we used the energy-based decision rule [15].

## 4 Discussion

In this paper, we have developed a new family of kernel functions that mimic the computation in large, multilayer neural nets. On challenging data sets, we have obtained results that outperform previous SVMs and compare favorably to deep belief nets. More significantly, our experiments validate the basic intuitions behind deep learning in the altogether different context of kernel-based architectures. A similar validation was provided by recent work on kernel methods for semi-supervised embedding [7]. We hope that our results inspire more work on kernel methods for deep learning.

There are many possible directions for future work. For SVMs, we are currently experimenting with arc-cosine kernel functions of fractional and (even negative) degree $n$. For MKMs, we are hoping to explore better schemes for feature selection [21, 22] and kernel selection [23]. Also, it would be desirable to incorporate prior knowledge, such as the invariances modeled by *convolutional* neural nets [24, 4], though it is not obvious how to do so. These issues and others are left for future work.

## A  Derivation of kernel function

In this appendix, we show how to evaluate the multidimensional integral in eq. (1) for the arc-cosine kernel. Let $\theta$ denote the angle between the inputs $\mathbf{x}$ and $\mathbf{y}$. Without loss of generality, we can take $\mathbf{x}$ to lie along the $w_1$ axis and $\mathbf{y}$ to lie in the $w_1 w_2$-plane. Integrating out the orthogonal coordinates of the weight vector $\mathbf{w}$, we obtain the result in eq. (3) where $J_n(\theta)$ is the remaining integral:

$$J_n(\theta) = \int dw_1\, dw_2\, e^{-\frac{1}{2}(w_1^2 + w_2^2)}\, \Theta(w_1)\, \Theta(w_1 \cos\theta + w_2 \sin\theta)\, w_1^n\, (w_1 \cos\theta + w_2 \sin\theta)^n. \quad (14)$$

Changing variables to $u = w_1$ and $v = w_1 \cos\theta + w_2 \sin\theta$, we simplify the domain of integration to the first quadrant of the $uv$-plane:

$$J_n(\theta) = \frac{1}{\sin\theta} \int_0^\infty du \int_0^\infty dv\, e^{-(u^2 + v^2 - 2uv\cos\theta)/(2\sin^2\theta)}\, u^n v^n. \quad (15)$$

The prefactor of $(\sin\theta)^{-1}$ in eq. (15) is due to the Jacobian. To simplify the integral further, we adopt polar coordinates $u = r\cos(\frac{\psi}{2} + \frac{\pi}{4})$ and $v = r\sin(\frac{\psi}{2} + \frac{\pi}{4})$. Then, integrating out the radius coordinate $r$, we obtain:

$$J_n(\theta) = n!\,(\sin\theta)^{2n+1} \int_0^{\frac{\pi}{2}} d\psi\, \frac{\cos^n\psi}{(1 - \cos\theta \cos\psi)^{n+1}}. \quad (16)$$

To evaluate eq. (16), we first consider the special case $n = 0$. The following result can be derived by contour integration in the complex plane [25]:

$$\int_0^{\pi/2} \frac{d\psi}{1 - \cos\theta \cos\psi} = \frac{\pi - \theta}{\sin\theta}. \quad (17)$$

Substituting eq. (17) into our expression for the angular part of the kernel function in eq. (16), we recover our earlier claim that $J_0(\theta) = \pi - \theta$. Related integrals for the special case $n = 0$ can also be found in earlier work [8]. For the case $n > 0$, the integral in eq. (16) can be performed by the method of differentiating under the integral sign. In particular, we note that:

$$\int_0^{\frac{\pi}{2}} d\psi\, \frac{\cos^n\psi}{(1 - \cos\theta \cos\psi)^{n+1}} = \frac{1}{n!} \frac{\partial^n}{\partial(\cos\theta)^n} \int_0^{\pi/2} \frac{d\psi}{1 - \cos\theta \cos\psi}. \quad (18)$$

Substituting eq. (18) into eq. (16), then appealing to the previous result in eq. (17), we recover the expression for $J_n(\theta)$ in eq. (4).

# References

[1] Y. Bengio and Y. LeCun. *Scaling learning algorithms towards AI*. MIT Press, 2007.

[2] G.E. Hinton, S. Osindero, and Y.W. Teh. A fast learning algorithm for deep belief nets. *Neural Computation*, 18(7):1527–1554, 2006.

[3] G.E. Hinton and R. Salakhutdinov. Reducing the dimensionality of data with neural networks. *Science*, 313(5786):504–507, July 2006.

[4] M.A. Ranzato, F.J. Huang, Y.L. Boureau, and Y. LeCun. Unsupervised learning of invariant feature hierarchies with applications to object recognition. In *Proceedings of the 2007 IEEE Conference on Computer Vision and Pattern Recognition (CVPR-07)*, pages 1–8, 2007.

[5] R. Collobert and J. Weston. A unified architecture for natural language processing: deep neural networks with multitask learning. In *Proceedings of the 25th International Conference on Machine Learning (ICML-08)*, pages 160–167, 2008.

[6] Y. Bengio. Learning deep architectures for AI. *Foundations and Trends in Machine Learning*, to appear, 2009.

[7] J. Weston, F. Ratle, and R. Collobert. Deep learning via semi-supervised embedding. In *Proceedings of the 25th International Conference on Machine Learning (ICML-08)*, pages 1168–1175, 2008.

[8] C.K.I. Williams. Computation with infinite neural networks. *Neural Computation*, 10(5):1203–1216, 1998.

[9] R.H.R. Hahnloser, H.S. Seung, and J.J. Slotine. Permitted and forbidden sets in symmetric threshold-linear networks. *Neural Computation*, 15(3):621–638, 2003.

[10] R.M. Neal. *Bayesian Learning for Neural Networks*. Springer-Verlag New York, Inc., 1996.

[11] H. Larochelle, D. Erhan, A. Courville, J. Bergstra, and Y. Bengio. An empirical evaluation of deep architectures on problems with many factors of variation. In *Proceedings of the 24th International Conference on Machine Learning (ICML-07)*, pages 473–480, 2007.

[12] C.C. Chang and C.J. Lin. *LIBSVM: a library for support vector machines*, 2001. Software available at `http://www.csie.ntu.edu.tw/~cjlin/libsvm`.

[13] B. Schölkopf, A. Smola, and K. Müller. Nonlinear component analysis as a kernel eigenvalue problem. *Neural Computation*, 10(5):1299–1319, 1998.

[14] I. Guyon and A. Elisseeff. An introduction to variable and feature selection. *Journal of Machine Learning Research*, 3:1157–1182, 2003.

[15] K.Q. Weinberger and L.K. Saul. Distance metric learning for large margin nearest neighbor classification. *Journal of Machine Learning Research*, 10:207–244, 2009.

[16] B. Schölkopf, A. J. Smola, and K.-R. Müller. Nonlinear component analysis as a kernel eigenvalue problem. Technical Report 44, Max-Planck-Institut für biologische Kybernetik, 1996.

[17] J. Goldberger, S. Roweis, G.E. Hinton, and R. Salakhutdinov. Neighbourhood components analysis. In L.K. Saul, Y. Weiss, and L. Bottou, editors, *Advances in Neural Information Processing Systems 17*, pages 513–520. MIT Press, 2005.

[18] Y. Bengio, P. Lamblin, D. Popovici, and H. Larochelle. Greedy layer-wise training of deep networks. In B. Schölkopf, J. Platt, and T. Hoffman, editors, *Advances in Neural Information Processing Systems 19*, pages 153–160. MIT Press, 2007.

[19] S. Chopra, R. Hadsell, and Y. LeCun. Learning a similarity metric discriminatively, with application to face verification. In *Proceedings of the 2005 IEEE Conference on Computer Vision and Pattern Recognition (CVPR-05)*, pages 539–546, 2005.

[20] Y. LeCun and C. Cortes. The MNIST database of handwritten digits. `http://yann.lecun.com/exdb/mnist/`.

[21] M. Tipping. Sparse kernel principal component analysis. In *Advances in Neural Information Processing Systems 13*. MIT Press, 2001.

[22] A.J. Smola, O.L. Mangasarian, and B. Schölkopf. Sparse kernel feature analysis. Technical Report 99-04, University of Wisconsin, Data Mining Institute, Madison, 1999.

[23] G. Lanckriet, N. Cristianini, P. Bartlett, L.E. Ghaoui, and M.I. Jordan. Learning the kernel matrix with semidefinite programming. *Journal of Machine Learning Research*, 5:27–72, 2004.

[24] Y. LeCun, B. Boser, J.S. Denker, D. Henderson, R.E. Howard, W. Hubbard, and L.D. Jackel. Backpropagation applied to handwritten zip code recognition. *Neural Computation*, 1(4):541–551, 1989.

[25] G.F. Carrier, M. Krook, and C.E. Pearson. *Functions of a Complex Variable: Theory and Technique*. Society for Industrial and Applied Mathematics, 2005.

